# Dirichlet-Enhanced Spam Filtering based on Biased Samples

**Steffen Bickel and Tobias Scheffer**
Max-Planck-Institut für Informatik, Saarbrücken, Germany
{bickel, scheffer}@mpi-inf.mpg.de

## Abstract

We study a setting that is motivated by the problem of filtering spam messages for many users. Each user receives messages according to an individual, unknown distribution, reflected only in the unlabeled inbox. The spam filter for a user is required to perform well with respect to this distribution. Labeled messages from publicly available sources can be utilized, but they are governed by a distinct distribution, not adequately representing most inboxes. We devise a method that minimizes a loss function with respect to a user's personal distribution based on the available biased sample. A nonparametric hierarchical Bayesian model furthermore generalizes across users by learning a common prior which is imposed on new email accounts. Empirically, we observe that bias-corrected learning outperforms naive reliance on the assumption of independent and identically distributed data; Dirichlet-enhanced generalization across users outperforms a single ("one size fits all") filter as well as independent filters for all users.

## 1 Introduction

Design and analysis of most machine learning algorithms are based on the assumption that the training data be drawn independently and from the same stationary distribution that the resulting model will be exposed to. In many application scenarios, however, control over the data generation process is less perfect, and so this *iid* assumption is often a naive over-simplification. In econometrics, learning from biased samples is a common phenomenon, where the willingness to respond to surveys is known to depend on several characteristics of the person queried; work that led to a method for correcting sample selection bias for a class of regression problems has been distinguished by a Nobel Prize [6]. In machine learning, the case of training data that is only biased with respect to the ratio of class labels has been studied [4, 7]. Zadrozny [14] has derived a bias correction theorem that applies when the bias is conditionally independent of the class label given the instance, and when every instance has a nonzero probability of being drawn into the sample. Sample bias correction for maximum entropy density estimation [3] and the analysis of the generalization error under covariate shift [12] follow the same intuition.

In our email spam filtering setting, a server handles many email accounts (in case of our industrial partner, several millions), and delivers millions of emails per day. A magnitude of spam and "ham" (*i.e.,* non-spam) sources are publicly available. They include collections of emails caught in "spam traps" – email addresses that are published on the web in an invisible font and are harvested by spammers [11] – the Enron corpus that was disclosed in the course of the Enron trial [8], and SpamAssassin data. These collections have diverse properties and none of them represents the global distribution of all emails, let alone the distribution received by some particular user. The resulting bias does not only hinder learning, but also leads to skewed accuracy estimates, since individuals may receive a larger proportion of emails that a filter classifies less confidently.

The following data generation model is paramount to our problem setting. An unknown process, characterized by a distribution $p(\theta_i|\beta)$, generates parameters $\theta_i$. The $\theta_i$ parameterize distributions $p(\mathbf{x}, y|\theta_i)$ over instances $\mathbf{x}$ (emails) and class labels $y$. Each $p(\mathbf{x}, y|\theta_i)$ corresponds to the $i$-th user's distribution of incoming spam ($y = +1$) or ham ($y = -1$) messages $\mathbf{x}$.

The goal is to obtain a classifier $f_i : \mathbf{x} \mapsto y$ for each $\theta_i$ that minimizes the expectation of some loss function $E_{(\mathbf{x},y)\sim\theta_i}[\ell(f(\mathbf{x}), y)]$, defined with respect to the (unknown) distribution $\theta_i$.

Labeled training data $L$ are drawn from a blend of data sources (public email archives), resulting in a density $p(\mathbf{x}, y|\lambda) = p(\mathbf{x}|\lambda)p(y|\mathbf{x}, \lambda)$ with parameter $\lambda$ that governs $L$. The relation between the $\theta_i$ and $\lambda$ is such that (a) any $\mathbf{x}$ that has nonzero probability density $p(\mathbf{x}|\lambda)$ of being drawn into the sample $L$ also has a nonzero probability $p(\mathbf{x}|\theta_i)$ under the target distributions $\theta_i$; and (b) the concept of spam is consensual for all users and the labeled data; *i.e.,* $p(y|\mathbf{x}, \lambda) = p(y|\mathbf{x}, \theta_i)$ for all users $i$.

In addition to the (nonempty) labeled sample, zero or more unlabeled data $U_i$ are available for each $\theta_i$ and are drawn according to $\theta_i$. The unlabeled sample $U_i$ is the inbox of user $i$. The inbox is empty for a newly established account and grows from there on. Our problem setting corresponds to an application scenario in which users are not prepared to manually tag spam messages in their inbox. Due to privacy and legal constraints, we are not allowed to personally read (or label) any single personal email; but the unlabeled messages may be used as input to an automated procedure.

The individual distributions $\theta_i$ are neither independent (identical spam messages are sent to many users), nor are they likely to be identical: distributions of inbound messages vary greatly between (professional, recreational, American, Chinese, . . . ) email users. We develop a nonparametric hierarchical Bayesian model that allows us to impose a common prior on new $\theta_i$. Such generalization may be particularly helpful for users with little or no available data $U_i$. The desired outcome of the learning process is an array of personalized spam filters for all users.

The rest of this paper is structured as follows. We devise our solution in Section 2. In Section 3, we study the effectiveness of correcting sample bias for spam, and of using a Dirichlet process to generalize across users, experimentally. Section 4 concludes.

## 2 Learning from Biased Data

The available labeled data $L$ are governed by $p(\mathbf{x}|\lambda)$; directly training a classifier on $L$ would therefore minimize the expected loss $E_{(\mathbf{x},y)\sim\lambda}[\ell(f(\mathbf{x}), y)]$ with respect to $p(\mathbf{x}|\lambda)$. By contrast, the task is to find classifiers $f_i$ that minimize, for user $i$, the expected loss $E_{(\mathbf{x},y)\sim\theta_i}[\ell(f(\mathbf{x}), y)]$ with respect to $p(\mathbf{x}|\theta_i)$. We can minimize the loss with respect to $\theta_i$ from a sample $L$ whose instances are governed by $\lambda$ when each instance is re-weighted. The weights have to be chosen such that minimizing the loss on the weighted sample $L$ amounts to minimizing the loss with respect to $\theta_i$.

In order to derive weighting factors with this property, consider the following model of the process that selects the labeled sample $L$. After drawing an instance $\mathbf{x}$ according to $p(\mathbf{x}|\theta_i)$, a coin $s$ is tossed with probability $p(s|\mathbf{x}, \theta_i, \lambda)$. We move $\mathbf{x}$ into the labeled sample (and add the proper class label) if $s = 1$; otherwise, $\mathbf{x}$ is discarded. Our previous assumption that any $\mathbf{x}$ with positive $p(\mathbf{x}|\lambda)$ also has a positive $p(\mathbf{x}|\theta_i)$ implies that there exists a $p(s|\mathbf{x}, \theta_i, \lambda)$ such that

$$p(\mathbf{x}|\lambda) \propto p(\mathbf{x}|\theta_i)p(s = 1|\mathbf{x}, \theta_i, \lambda). \tag{1}$$

That is, repeatedly executing the above process with an appropriate $p(s|\mathbf{x}, \theta_i, \lambda)$ will create a sample of instances governed by $p(\mathbf{x}|\lambda)$. Equation 1 defines $p(s|\mathbf{x}, \theta_i, \lambda)$; the succeeding subsections will be dedicated to estimating it from the available data. Since $p(s|\mathbf{x}, \theta_i, \lambda)$ describes the discrepancy between the sample distribution $p(\mathbf{x}|\lambda)$ and the target $p(\mathbf{x}|\theta_i)$, we refer to it as the *sample bias*.

But let us first show that minimizing the loss on $L$ with instances re-weighted by $p(s|\mathbf{x}, \theta_i, \lambda)^{-1}$ in fact minimizes the expected loss with respect to $\theta_i$. The rationale behind this claim deviates only in minor points from the proof of the bias correction theorem of [14]. Proposition 1 introduces a normalizing constant $p(s = 1|\theta_i, \lambda)$. Its value can be easily obtained as it normalizes Equation 1.

**Proposition 1** *The expected loss with respect to* $p(\mathbf{x}, y|\bar{\theta}_i) = p(\mathbf{x}, y|\lambda)\frac{p(s=1|\theta_i,\lambda)}{p(s=1|\mathbf{x},\theta_i,\lambda)}$ *equals the expected loss with respect to* $p(\mathbf{x}, y|\theta_i)$, *when* $p(s|\mathbf{x}, \theta_i, \lambda)$ *satisfies Equation 1.*

**Proof.** Equation 2 expands the expected value and the definition of $p(\mathbf{x}, y|\bar{\theta}_i)$ in Proposition 1. Equation 3 splits $p(\mathbf{x}, y|\lambda)$. We apply the definition of $p(s|\mathbf{x}, \theta_i, \lambda)$ (Equation 1) and obtain Equation 4. Equation 4 is rewritten as an expected value.

$$E_{(\mathbf{x},y)\sim\bar{\theta}_i}[\ell(f(\mathbf{x}), y)] \quad = \quad \int \ell(f(\mathbf{x}), y)p(\mathbf{x}, y|\lambda)\frac{p(s = 1|\theta_i, \lambda)}{p(s = 1|\mathbf{x}, \theta_i, \lambda)}d(\mathbf{x}, y) \tag{2}$$

$$= \quad \int \ell(f(\mathbf{x}), y)p(y|\mathbf{x}, \lambda)p(\mathbf{x}|\lambda)\frac{p(s = |\theta_i, \lambda)}{p(s = 1|\mathbf{x}, \theta_i, \lambda)}d(\mathbf{x}, y) \tag{3}$$

$$= \quad \int \ell(f(\mathbf{x}), y)p(y|\mathbf{x}, \theta_i)p(\mathbf{x}|\theta_i)d(\mathbf{x}, y) = E_{(\mathbf{x},y)\sim\theta_i}[\ell(f(\mathbf{x}), y)] \tag{4}$$

## 2.1 Individualized Bias Estimation

Equation 1 says that there is an unknown $p(s|\mathbf{x}, \theta_i, \lambda)$ with $p(\mathbf{x}|\lambda) \propto p(\mathbf{x}|\theta_i)p(s = 1|\mathbf{x}, \theta_i, \lambda)$ which we call the sample bias. We will now discuss how to obtain an estimate $\hat{p}_I(s|\mathbf{x}, \theta_i, \lambda)$. The *individualized empirical sample bias* is an estimate of the unknown true bias, conditioned on a user's unlabeled inbox $U_i$ and labeled data $L$; hence, $\hat{p}_I(s|\mathbf{x}, \theta_i, \lambda) = p(s|\mathbf{x}, U_i, L)$.

Equation 1 immediately implies

$$p(s = 1|\mathbf{x}, \theta_i, \lambda) \propto \frac{p(\mathbf{x}|\lambda)}{p(\mathbf{x}|\theta_i)}, \tag{5}$$

but neither $p(\mathbf{x}|\lambda)$ nor $p(\mathbf{x}|\theta_i)$ are known. However, distribution $p(\mathbf{x}|\lambda)$ is reflected in the labeled sample $L$, and distribution $p(\mathbf{x}|\theta_i)$ in the unlabeled inbox $U_i$. Instances in $L$ are examples that have been selected into the labeled sample; *i.e.,* $s = 1|\mathbf{x} \in L$. Instances in $U_i$ have not been selected into the labeled sample; *i.e.,* $s = 0|\mathbf{x} \in U_i$. We define $\mathbf{s}_{U_i,L}$ to be the vector of selection decisions for all instances in $U_i$ and $L$. That is, $\mathbf{s}_{U_i,L}$ contains $|U_i|$ elements that are 0, and $|L|$ elements that are 1.

A density estimator $\hat{p}(s|\mathbf{x}, \lambda, \theta_i)$ can be trained on the instances in $L$ and $U_i$, using vector $\mathbf{s}_{U_i,L}$ as target variable. We use a regularized logistic regression density estimator parameterized with $\mathbf{w}_i$:

$$\hat{p}_I(s = 1|\mathbf{x}, \lambda, \theta_i) = p(s = 1|\mathbf{x}; \mathbf{w}_i) = \frac{1}{1 + e^{\langle\mathbf{w}_i, \mathbf{x}\rangle}}. \tag{6}$$

The likelihood of the density estimator is

$$P(\mathbf{s}_{U_i,L}|\mathbf{w}, U_i, L) = \prod_{\mathbf{x}_u \in U_i} p(s = 0|\mathbf{x}_u, \mathbf{w}) \prod_{\mathbf{x}_\ell \in L} p(s = 1|\mathbf{x}_\ell, \mathbf{w}). \tag{7}$$

We train parameters $\mathbf{w}_i = \operatorname{argmax}_\mathbf{w} \log P(\mathbf{s}_{U_i,L}|\mathbf{w}, U_i, L) + \log \eta(\mathbf{w})$ (we write $\eta(\mathbf{w})$ for the regularizer) [15] using the fast implementation of regularized logistic regression of [9].

## 2.2 Dirichlet-Enhanced Bias Estimation

This section addresses estimation of the sample bias $p(s|\mathbf{x}, \theta_{n+1}, \lambda)$ for a new user $n+1$ by generalizing across existing users $U_1, \ldots, U_n$. The resulting estimate $\hat{p}_D(s|\mathbf{x}, \theta_{n+1}, \lambda)$ will be conditioned on the new user's inbox $U_{n+1}$ and the labeled data $L$, but also on all other users' inboxes. We write $\hat{p}_D(s|\mathbf{x}, \theta_{n+1}, \lambda) = p(s|\mathbf{x}, U_{n+1}; L, U_1, \ldots, U_n)$ for the *Dirichlet-enhanced empirical sample bias*.

Equation 1 says that there is a $p(s = 1|\mathbf{x}, \theta_{n+1}, \lambda)$ for user $n + 1$ that satisfies Equation 5. Let us assume a parametric form (we employ a logistic model), and let $\mathbf{w}_{n+1}$ be the parameters that satisfy $p(s = 1|\mathbf{x}, \theta_{n+1}, \lambda) = p(s = 1|\mathbf{x}; \mathbf{w}_{n+1}) \propto p(\mathbf{x}|\lambda)/p(\mathbf{x}|\theta_{n+1})$. We resort to a Dirichlet process (DP) [5] $G(\mathbf{w}_i)$ as a model for the prior belief on $\mathbf{w}_{n+1}$ given $\mathbf{w}_1, \ldots, \mathbf{w}_n$. Dirichlet process $G|\{\alpha, G_0\} \sim DP(\alpha, G_0)$ with concentration parameter $\alpha$ and base distribution $G_0$ generates parameters $\mathbf{w}_i$: The first element $\mathbf{w}_1$ is drawn according to $G_0$; in our case, the uninformed prior. It generates $\mathbf{w}_{n+1}$ according to Equation 8, where $\delta(\mathbf{w}_i)$ is a point distribution centered at $\mathbf{w}_i$.

$$\mathbf{w}_{n+1}|\mathbf{w}_1, \ldots, \mathbf{w}_n \sim \frac{\alpha G_0 + \sum_{i=1}^n \delta(\mathbf{w}_i)}{\alpha + n} \tag{8}$$

Equation 9 integrates over the parameter of the bias for new user $n + 1$. Equation 10 splits the posterior into the likelihood of the sample selection coin tosses and the common prior which is

modeled as a Dirichlet process.

$$p(s|\mathbf{x}, U_{n+1}; L, U_1, \ldots, U_n) = \int p(s|\mathbf{x}; \mathbf{w}) p(\mathbf{w}|U_{n+1}; L, U_1, \ldots, U_n) d\mathbf{w} \tag{9}$$

$$p(\mathbf{w}|U_{n+1}, L, U_1, \ldots, U_n) \propto P(\mathbf{s}_{U_{n+1},L}|\mathbf{w}, U_{n+1}, L)\hat{G}(\mathbf{w}|L, U_1, \ldots, U_n) \tag{10}$$

Likelihood $P(\mathbf{s}_{U_{n+1},L}|\mathbf{w}, U_{n+1}, L)$ is resolved in Equation 7 for a logistic model of the bias.

## 2.3 Estimation of the Dirichlet Process

The parameters of previous users' bias $\mathbf{w}_1, \ldots, \mathbf{w}_n$ constitute the prior $\mathbf{w}_{n+1}|\{\mathbf{w}_i\}_{i=1}^n \sim G$ for user $n + 1$. Since the parameters $\mathbf{w}_i$ are not observable, an estimate $\mathbf{w}_{n+1}|L, \{U_i\}_{i=1}^n \sim \hat{G}$ has to be based on the available data. Exact calculation of this prior requires integrating over the $\mathbf{w}_1, \ldots, \mathbf{w}_n$; since this is not feasible, MCMC sampling [10] or variational approximation [1] can be used.

In our application, the model of $p(s|\mathbf{x}, \theta_i, \lambda)$ involves a regularized logistic regression in a space of more than 800,000 dimensions. In each iteration of the MCMC process or the variational inference of [1], logistic density estimators for all users would need to be trained—which is prohibitive. We therefore follow [13] and approximate the Dirichlet Process as

$$\hat{G}(\mathbf{w}) \approx \frac{\alpha G_0 + \sum_{i=1}^n \phi_i \delta(\mathbf{w}_i^*)}{\alpha + n}. \tag{11}$$

Compared to the original Equation 8, the sum of point distributions at true parameters $\mathbf{w}_i$ is replaced by a weighted sum over point distributions at pivotal $\mathbf{w}_i^*$. Parameter estimation is divided in two steps. First, pivotal models of the sample bias are trained for each user $i$, solely based on a user's inbox and the labeled data. Secondly, parameters $\phi_i$ are estimated using variational EM; they express correlations between, and allow for generalization across, multiple users. Tresp and Yu [13] suggest to use a maximum likelihood estimate $\mathbf{w}_i^*$; we implement $\mathbf{w}_i^*$ by training logistic regression models

$$p(s = 1|\mathbf{x}; \mathbf{w}_i^*) = \frac{1}{1 + e^{\langle \mathbf{w}_i^*, \mathbf{x}\rangle}}$$
$$\text{with } \mathbf{w}_i^* = \text{argmax}_\mathbf{w} \log P(\mathbf{s}_{U_i, L}|\mathbf{w}, U_i, L) + \log \eta(\mathbf{w}). \tag{12}$$

Algorithmically, the pivotal models are obtained analogously to the individualized estimation of the selection bias for each user described in Section 2.1.

After the pivotal models have been identified, an EM algorithm maximizes the likelihood over the parameters $\phi_i$. For the E step we rely on the assumption that the posterior is a weighted sum over point distributions at the pivotal density estimates (Equation 13). With this assumption, the posterior is no longer a continuous distribution and the E step resolves to the computation of a discrete number of variational parameters $\phi_{ij}$ (Equation 14).

$$\hat{p}(\mathbf{w}|U_j, L) = \sum_{i=1}^n \phi_{ij}\delta(\mathbf{w}_i^*) \tag{13}$$

$$\phi_{ij} \propto P(\mathbf{s}_{U_j,L}|\mathbf{w}^*, U_j, L)\hat{G}(\mathbf{w}_i^*) \tag{14}$$

Equation 11 yields the M step with $\phi_i = \sum_{j=1}^n \phi_{ij}$. Likelihood $P(\mathbf{s}_{U_j,L}|\mathbf{w}^*, U_j, L)$, is calculated as in Equation 7. The entire estimation procedure is detailed in Table 1, steps 1 through 3.

## 2.4 Inference

Having obtained pivotal models $p(s|\mathbf{x}; \mathbf{w}_i^*)$ and parameters $\phi_i$, we need to infer the Dirichlet-enhanced empirical sample bias $p(s|\mathbf{x}, U_i; L, U_1, \ldots, U_n)$. During the training procedure, $i$ is one of the known users from $U_1, \ldots, U_n$. At application time, we may furthermore experience a message bound for user $n + 1$.

Without loss of generality, we discuss the inference problem for a new user $n + 1$. Inserting $\hat{G}(\mathbf{w})$ into Eqs. 9 and 10 leads to Equation 15. Expanding $\hat{G}(\mathbf{w})$ according to Eq. 11 yields Equation 16.

$$p(s|\mathbf{x}, U_{n+1}; L, U_1, \ldots, U_n) \propto \int p(s|\mathbf{x}; \mathbf{w})P(\mathbf{s}_{U_{n+1},L}|\mathbf{w}, U_{n+1}, L)\hat{G}(\mathbf{w})d\mathbf{w} \tag{15}$$

$$\propto \alpha \int p(s|\mathbf{x}; \mathbf{w})P(\mathbf{s}_{U_{n+1},L}|\mathbf{w}, U_{n+1}, L)G_0(\mathbf{w})d\mathbf{w} \tag{16}$$

$$+ \sum_{i=1}^n p(s|\mathbf{x}; \mathbf{w}_i^*)P(\mathbf{s}_{U_{n+1},L}|\mathbf{w}_i^*, U_{n+1}, L)\phi_i$$

The second summand in Equation 16 is determined by summing over the pivotal models $p(s|\mathbf{x}; \mathbf{w}_i^*)$. The first summand can be determined by applying Bayes' rule in Equation 17; $G_0$ is the uninformed prior; the resulting term $p(s|\mathbf{x}, U_{n+1}, L) = p(s|\mathbf{x}; \mathbf{w}_{n+1}^*)$ is the outcome of a new pivotal density estimator, trained to discriminate $L$ against $U_{n+1}$. It is determined as in Equation 12.

$$\int p(s|\mathbf{x}; \mathbf{w})P(\mathbf{s}_{U_{n+1},L}|\mathbf{w}, U_{n+1}, L)G_0(\mathbf{w})d\mathbf{w} \quad \propto \quad \int p(s|\mathbf{x}; \mathbf{w})p(\mathbf{w}|U_{n+1}, L)d\mathbf{w} \quad (17)$$

$$= \quad p(s|\mathbf{x}, U_{n+1}, L) \quad (18)$$

The Dirichlet-enhanced empirical sample bias $p(s|\mathbf{x}, U_{n+1}; L, U_1, \ldots, U_n)$ for user $n+1$ is a weighted sum of the pivotal density estimate $p(s|\mathbf{x}; \mathbf{w}_{n+1}^*)$ for user $n+1$, and models $p(s|\mathbf{x}; \mathbf{w}_i^*)$ of all users $i$; the latter are weighted according to their likelihood $P(\mathbf{s}_{U_{n+1},L}|\mathbf{w}_i^*, U_{n+1}, L)$ of observing the messages of user $n+1$. Inference for the users that are available at training time is carried out in step 4(a) of the training procedure (Table 1).

Table 1: Dirichlet-enhanced, bias-corrected spam filtering.

---

**Input:** Labeled data $L$, unlabeled inboxes $U_1, \ldots, U_n$.

1. **For** all users $i = 1 \ldots n$: Train a pivotal density estimator $\hat{p}(s=1|\mathbf{x}, \mathbf{w}_i^*)$ as in Eq. 12.

2. **Initialize** $\hat{G}^0(\mathbf{w}_i^*)$ by setting $\phi_i = 1$ for $i = 1 \ldots n$.

3. **For** $t = 1, \ldots$ **until** convergence:

   (a) **E-step: For** all $i$, $j$, estimate $\phi_{ij}^t$ from Equation 14 using $\hat{G}^{t-1}$ and the density estimators $p(s|\mathbf{x}, \mathbf{w}_i^*)$.

   (b) **M-step:** Estimate $\hat{G}^t(\mathbf{w}_i^*)$ according to Equation 11 using $\phi_i = \sum_{j=1}^{n} \phi_{ij}^t$.

4. **For** all users $i$:

   (a) **For** all $\mathbf{x} \in L$: determine empirical sample bias $p(s|\mathbf{x}, U_i; L, U_1, \ldots, U_n)$, conditioned on the observables according to Equation 16.

   (b) Train SVM classifier $f_i : \mathcal{X} \to \{\text{spam}, \text{ham}\}$ by solving Optimization Problem 1.

**Return** classifiers $f_i$ for all users $i$.

---

## 2.5 Training a Bias-Corrected Support Vector Machine

Given the requirement of high accuracy and the need to handle many attributes, SVMs are widely acknowledged to be a good learning mechanism for spam filtering [2]. The final bias-corrected SVM $f_{n+1}$ can be trained by re-sampling or re-weighting $L$ according to $s(\mathbf{x}) = \frac{p(s=1|\theta_i, \lambda)}{p(s=1|\mathbf{x}, U_{n+1}; L, U_1, \ldots, U_n)}$, where $p(s|\mathbf{x}, U_{n+1}; L, U_1, \ldots, U_n)$ is the empirical sample bias and $p(s=1|\theta_i, \lambda)$ is the normalizer that assures $\sum_{\mathbf{x} \in L} s(\mathbf{x}) = |L|$. Let $\mathbf{x}_k \in L$ be an example that incurs a margin violation (*i.e.*, slack term) of $\xi_k$. The expected contribution of $\mathbf{x}_k$ to the SVM criterion is $s(\mathbf{x})\xi_k$ because $\mathbf{x}_k$ will be drawn $s(\mathbf{x})$ times on average into each re-sampled data set. Therefore, training the SVM on the re-sampled data or optimizing with re-scaled slack terms lead to identical optimization problems.

**Optimization Problem 1** *Given labeled data $L$, re-sampling weights $s(\mathbf{x})$, and regularization parameter $C$; over all $\mathbf{v}$, $b$, $\xi_1, \ldots, \xi_m$, minimize*

$$\frac{1}{2}|\mathbf{v}|^2 + C\sum_{k=1}^{m} s(\mathbf{x})\xi_k \quad (19)$$

$$\textit{subject to } \forall_{k=1}^{m} y_k(\langle \mathbf{v}, \mathbf{x}_k \rangle + b) \geq 1 - \xi_k; \quad \forall_{k=1}^{m} \xi_k \geq 0. \quad (20)$$

The bias-corrected spam filter is trained in step 4(b) of the algorithm (Table 1).

## 2.6 Incremental Update

The Dirichlet-enhanced bias correction procedure is intrinsically incremental, which fits into the typical application scenario. When a new user $n+1$ subscribes to the email service, the prior

Table 2: Email accounts used for experimentation.

| User | Ham | Spam |
|---|---|---|
| Williams | Enron/Williams | Dornbos spam trap (www.dornbos.com) (part 1) |
| Beck | Enron/Beck | spam trap of Bruce Guenter (www.em.ca/∼bruceg/spam) |
| Farmer | Enron/Farmer | personal spam of Paul Wouters (www.xtdnet.nl/paul/spam) |
| Kaminski | Enron/Kaminski | spam collection of SpamArchive.org (part 1) |
| Kitchen | Enron/Kitchen | personal spam of the second author. |
| Lokay | Enron/Lokay | spam collection of SpamAssassin (www.spamassassin.org) |
| Sanders | Enron/Sanders | personal spam of Richard Jones (www.annexia.org/spam) |
| German traveler | Usenet/de.rec.reisen.misc | Dornbos spam trap (www.dornbos.com) (part 2) |
| German architect | Usenet/de.sci.architektur | spam collection of SpamArchive.org (part 2) |

$\mathbf{w}_{n+1}|L, \{U_i\}_{i=1}^n \sim \hat{G}$ is already available. A pivotal model $p(s|\mathbf{x}, U_{n+1}; L)$ can be trained; when $U_{n+1}$ is still empty (the new user has not yet received emails), then the regularizer of the density estimate $p(s|\mathbf{x}, U_{n+1}, L)$ resolves to the uniform distribution. Inference of $p(s|\mathbf{x}, U_{n+1}; L, U_1, \ldots, U_n)$ for the new user proceeds as discussed in Section 2.4.

When data $U_{n+1}$ becomes available, the prior can be updated. This update is exercised by invoking the EM estimation procedure with additional parameters $\theta_{n+1}^*$ and $\phi_{(n+1)}$. The estimates of $P(s_{U_j,L}|\mathbf{w}_i^*, U_j, L)$ for all pairs of existing users $i$ and $j$ do not change and can be reused. The EM procedure returns the updated prior $\mathbf{w}_{n+2}|L, \{U_i\}_{i=1}^{n+1} \sim \hat{G}$ for the next new user $n+2$.

## 3   Experiments

In our experiments, we study the relative benefit of the following filters. The baseline is constituted by a filter that is trained under *iid* assumption from the labeled data. The second candidate is a "one size fits all" bias-corrected filter. Here, all users' messages are pooled as unlabeled data and the bias $p(s|\mathbf{x}, \theta_{n+1}, \lambda)$ is modeled by an estimator $\hat{p}_O(s|\mathbf{x}, \theta_{n+1}, \lambda) = p(s|\mathbf{x}, \bigcup_{i=1}^{n+1} U_i, L)$. An individually bias-corrected filter uses estimators $\hat{p}_I(s|\mathbf{x}, \theta_{n+1}, \lambda) = p(s|\mathbf{x}, U_{n+1}, L)$. Finally, we assess the Dirichlet-enhanced bias-corrected filter. It uses the hierarchical Bayesian model to determine the empirical bias $\hat{p}_D(s|\mathbf{x}, \theta_{n+1}, \lambda) = p(s|\mathbf{x}, U_{n+1}; L, U_1, \ldots, U_n)$ conditioned on the new user's messages, the labeled data, and all previous users' messages.

Evaluating the filters with respect to the personal distributions of messages requires labeled emails from distinct users. We construct nine accounts using real but disclosed messages. Seven of them contain ham emails received by distinct Enron employees from the Enron corpus [8]; we use the individuals with the largest numbers of messages from a set of mails that have been cleaned from spam. We simulate two foreign users: the "German traveler" receives postings to a moderated German traveling newsgroup, the "German architect" postings to a newsgroup on architecture.

Each account is augmented with between 2551 and 6530 spam messages from a distinct source, see Table 2. The number of ham emails varies between 1189 and 5983, reflecting about natural ham-to-spam ratios. The ham section of the labeled data $L$ contains 4000 ham emails from the Spam-Assassin corpus, 1000 newsletters and 500 emails from Enron employee Taylor. The labeled data contain 5000 spam emails relayed by blacklisted servers. The data are available from the authors.

The total of 76,214 messages are transformed into binary term occurrence vectors with a total of 834,661 attributes; charset and base64 decoding are applied, email headers are discarded, tokens occurring less than 4 times are removed. SVM parameter $C$, concentration parameter $\alpha$, and the regularization parameter of the logistic regression are adjusted on a small reserved tuning set.

We iterate over all users and let each one play the role of the new user $n+1$. We then iterate over the size of the new user's inbox and average 10 repetitions of the evaluation process, sampling $U_{n+1}$ from the inbox and using the remaining messages as hold-out data for performance evaluation. We train the different filters on identical samples and measure the area under the ROC curve (AUC).

Figure 1 shows the AUC performance of the *iid* baseline and the three bias-corrected filters for the first two Enron and one of the German users. Error bars indicate standard error of the difference to

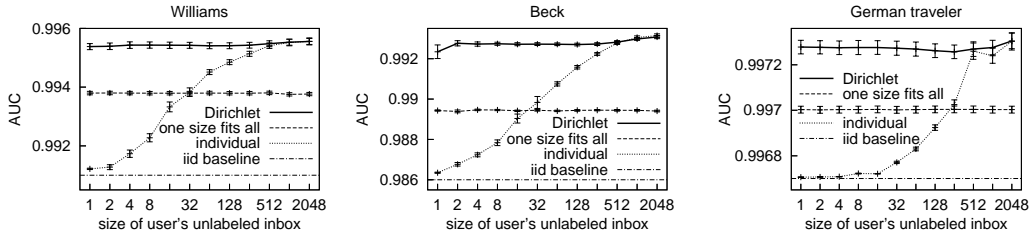

Figure 1: AUC of the *iid* baseline and the three bias-corrected filters versus size of $|U_{n+1}|$.

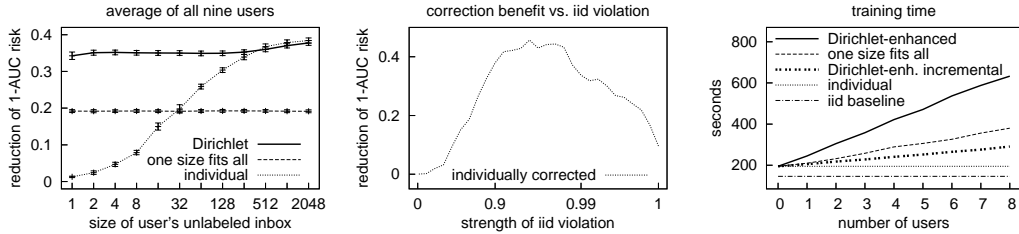

Figure 2: Average reduction of 1-AUC risk over all nine users (left); reduction of 1-AUC risk dependent on strength of *iid* violation (center); number of existing users vs. training time (right).

the *iid* filter. Figure 2 (left) aggregates the results over all nine users by averaging the rate by which the risk $1 - AUC$ is reduced. We compute this reduction as $1 - \frac{1 - AUC_{corrected}}{1 - AUC_{baseline}}$, where $AUC_{corrected}$ is one of the bias-corrected filters and $AUC_{baseline}$ is the AUC of the *iid* filter.

The benefit of the individualized bias correction depends on the number of emails available for that user; the $1 - AUC$ risk is reduced by 35-40% when many emails are available. The "one size fits all" filter is almost independent of the number of emails of the new user. On average, the Dirichlet-enhanced filter reduces the risk $1 - AUC$ by about 35% for a newly created account and by almost 40% when many personal emails have arrived. It outperforms the "one size fits all" filter even for an empty $U_{n+1}$ because fringe accounts (*e.g.,* the German users) can receive a lower weight in the common prior. The baseline $AUC$ of over 0.99 is typical for server-sided spam filtering; a 40% risk reduction that yields an AUC of 0.994 is still a very significant improvement of the filter that can be spent on a substantial reduction of the false positive rate, or on a higher rate of spam recognition.

The question occurs how strong a violation of the *iid* assumption the bias correction techniques can compensate. In order to investigate, we control the violation of the *iid* property of the labeled data as follows. We create a strongly biased sample by using only Enron users as test accounts $\theta_i$, and not using any Enron emails in the labeled data. We vary the proportion of strongly biased data versus randomly drawn Enron mails in the labeled training data (no email occurs in the training and testing data at the same time). When this proportion is zero, the labeled sample is drawn *iid* from the testing distributions; when it reaches 1, the sample is strongly biased. In Figure 2 (center) we observe that, averaged over all users, bias-correction is effective when the *iid* violation lies in a mid-range. It becomes less effective when the sample violates the *iid* assumption too strongly. In this case, "gaps" occur in $\lambda$; *i.e.,* there are regions that have zero probability in the labeled data $L \sim \lambda$ but nonzero probability in the testing data $U_i \sim \theta_i$. Such gaps render schemes that aim at reconstructing $p(\mathbf{x}|\theta_i)$ by weighting data drawn according to $p(\mathbf{x}|\lambda)$ ineffective.

Figure 2 (right) displays the total training time over the number of users. We fix $|U_{n+1}|$ to 16 and vary the number of users that influence the prior. The *iid* baseline and the individually corrected filter scale constantly. The Dirichlet-enhanced filter scales linearly in the number of users that constitute the common prior; the EM algorithm with a quadratic complexity in the number of users contributes only marginally to the training time. The training time is dominated by the training of the pivotal models (linear complexity). The Dirichlet enhanced filter with incremental update scales favorably compared to the "one size fits all" filter. Figure 2 is limited to the 9 accounts that we have engineered; the execution time is in the order of minutes and allows to handle larger numbers of accounts.

# 4 Conclusion

It is most natural to define the quality criterion of an email spam filter with respect to the distribution that governs the personal emails of its user. It is desirable to utilize available labeled email data, but assuming that these data were governed by the same distribution unduly over-simplifies the problem setting. Training a density estimator to characterize the difference between the labeled training data and the unlabeled inbox of a user, and using this estimator to compensate for this discrepancy, improves the performance of a personalized spam filter—provided that the inbox contains sufficiently many messages. Pooling the unlabeled inboxes of a group of users, training a density estimator on this pooled data, and using this estimator to compensate for the bias outperforms the individualized bias-correction only when very few unlabeled data for the new user are available.

We developed a hierarchical Bayesian framework which uses a Dirichlet process to model the common prior for a group of users. The Dirichlet-enhanced bias correction method estimates – and compensates for – the discrepancy between labeled training and unlabeled personal messages, learning from the new user's unlabeled inbox as well as from data of other users. Empirically, with a 35% reduction of the $1 - AUC$ risk for a newly created account, the Dirichlet-enhanced filter outperforms all other methods. When many unlabeled personal emails are available, both individualized and Dirichlet-enhanced bias correction reduce the $1 - AUC$ risk by nearly 40% on average.

### Acknowledgment

This work has been supported by Strato Rechenzentrum AG and by the German Science Foundation DFG under grant SCHE540/10-2.

# References

[1] D. Blei and M. Jordan. Variational methods for the Dirichlet process. In *Proceedings of the International Conference on Machine Learning*, 2004.

[2] H. Drucker, D. Wu, and V. Vapnik. Support vector machines for spam categorization. *IEEE Transactions on Neural Networks*, 10(5):1048–1055, 1999.

[3] M. Dudik, R. Schapire, and S. Phillips. Correcting sample selection bias in maximum entropy density estimation. In *Advances in Neural Information Processing Systems*, 2005.

[4] C. Elkan. The foundations of cost-sensitive learning. In *Proceedings of the International Joint Conference on Artificial Intellligence*, 2001.

[5] T. Ferguson. A Bayesian analysis of some nonparametric problems. *Annals of Statistics*, 1:209–230, 1973.

[6] J. Heckman. Sample selection bias as a specification error. *Econometrica*, 47:153–161, 1979.

[7] N. Japkowicz and S. Stephen. The class imbalance problem: A systematic study. *Intelligent Data Analysis*, 6:429–449, 2002.

[8] Bryan Klimt and Yiming Yang. The enron corpus: A new dataset for email classification research. In *Proceedings of the European Conference on Machine Learning*, 2004.

[9] P. Komarek. *Logistic Regression for Data Mining and High-Dimensional Classification*. Doctoral dissertation, Carnegie Mellon University, 2004.

[10] R. Neal. Markov chain sampling methods for Dirichlet process mixture models. *Journal of Computational and Graphical Statistics*, 9:249–265, 2000.

[11] Matthew Prince, Benjamin Dahl, Lee Holloway, Arthur Kellera, and Eric Langheinrich. Understanding how spammers steal your e-mail address: An analysis of the first six months of data from project honey pot. In *Proceedings of the Conference on Email and Anti-Spam*, 2005.

[12] M. Sugiyama and K.-R. Müller. Model selection under covariate shift. In *Proceedings of the International Conference on Artificial Neural Networks*, 2005.

[13] Volker Tresp and Kai Yu. An introduction to nonparametric hierarchical Bayesian modelling with a focus on multi-agent learning. In *Switching and Learning in Feedback Systems*, volume 3355 of *Lecture Notes in Computer Science*, pages 290–312. Springer, 2004.

[14] Bianca Zadrozny. Learning and evaluating classifiers under sample selection bias. In *Proceedings of the International Conference on Machine Learning*, 2004.

[15] T. Zhang and F. Oles. Text categorization based on regularized linear classifiers. *Information Retrieval*, 4(1):5–31, 2001.
